# Estimating analogical similarity by dot-products of Holographic Reduced Representations.

**Tony A. Plate**
Department of Computer Science, University of Toronto
Toronto, Ontario, Canada M5S 1A4
email: tap@ai.utoronto.ca

## Abstract

Models of analog retrieval require a computationally cheap method of estimating similarity between a probe and the candidates in a large pool of memory items. The vector dot-product operation would be ideal for this purpose if it were possible to encode complex structures as vector representations in such a way that the superficial similarity of vector representations reflected underlying structural similarity. This paper describes how such an encoding is provided by Holographic Reduced Representations (HRRs), which are a method for encoding nested relational structures as fixed-width distributed representations. The conditions under which structural similarity is reflected in the dot-product rankings of HRRs are discussed.

## 1 INTRODUCTION

Gentner and Markman (1992) suggested that the ability to deal with analogy will be a "Watershed or Waterloo" for connectionist models. They identified "structural alignment" as the central aspect of analogy making. They noted the apparent ease with which people can perform structural alignment in a wide variety of tasks and were pessimistic about the prospects for the development of a distributed connectionist model that could be useful in performing structural alignment.

In this paper I describe how Holographic Reduced Representations (HRRs) (Plate, 1991; Plate, 1994), a fixed-width distributed representation for nested structures, can be used to obtain fast estimates of analogical similarity. A HRR is a high dimensional vector,

and the vector dot-product of two HRRs is an efficiently computable estimate of the overall similarity between the two structures represented. This estimate reflects both surface similarity and some aspects of structural similarity,[1] even though alignments are not explicitly calculated. I also describe contextualization, an enrichment of HRRs designed to make dot-product comparisons of HRRs more sensitive to structural similarity.

## 2  STRUCTURAL ALIGNMENT & ANALOGICAL REMINDING

People appear to perform structural alignment in a wide variety of tasks, including perception, problem solving, and memory recall (Gentner and Markman, 1992; Markman, Gentner and Wisniewski, 1993). One task many researchers have investigated is analog recall. A subject is shown a number of stories and later is shown a probe story. The task is to recall stories that are similar to the probe story (and sometimes evaluate the degree of similarity and perform analogical reasoning).

MAC/FAC, a computer models of this process, has two stages(Gentner and Forbus, 1991). The first stage selects a few likely analogs from a large number of potential analogs. The second stage searches for an optimal (or at least good) mapping between each selected story and the probe story and outputs those with the best mappings. Two stages are necessary because it is too computationally expensive to search for an optimal mapping between the probe and all stories in memory. An important requirement for a first stage is that its performance scale well with both the size and number of episodes in long-term memory. This prevents the first stage of MAC/FAC from considering any structural features.

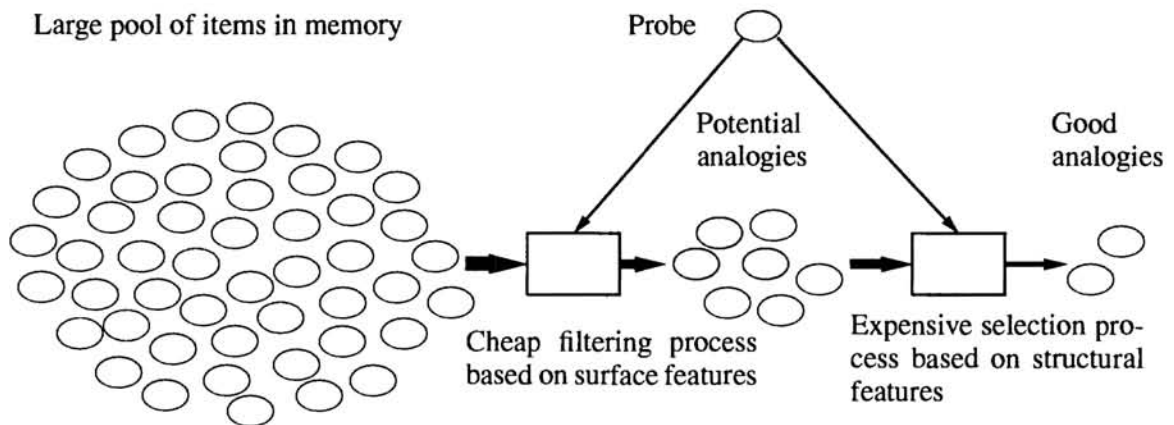

Figure 1: General architecture of a two-stage retrieval model.

While it is indisputable that people take structural correspondences into account when evaluating and using analogies (Gentner, Rattermann and Forbus, 1993), it is less certain whether structural similarity influences access to long term memory (i.e., the first-stage reminding process). Some studies have found little effect of analogical similarity on reminding (Gentner and Forbus, 1991; Gentner, Rattermann and Forbus, 1993), while others have found some effect (Wharton et al., 1994).

In any case, surface features appear to influence the likelihood of a reminding far more than do structural features. Studies that have found an effect of structural similarity on reminding seem to indicate the effect only exists, or is greater, in the presence of surface similarity (Gentner and Forbus, 1991; Gentner, Rattermann and Forbus, 1993; Thagard et al., 1990).

## 2.1  EXAMPLES OF ANALOGY BETWEEN NESTED STRUCTURES.

To test how well the HRR dot-product works as an estimate of analogical similarity between nested relational structures I used the following set of simple episodes (see Plate (1993) for the full set). The memorized episodes are similar in different ways to the probe. These examples are adapted from (Thagard et al., 1990).

| Probe: | | Spot bit Jane, causing Jane to flee from Spot. |
|---|---|---|
| Episodes in long-term memory: | | |
| E1 | (LS) | Fido bit John, causing John to flee from Fido. |
| E2 | $(AN^{cm})$ | Fred bit Rover, causing Rover to flee from Fred. |
| E3 | (AN) | Felix bit Mort, causing Mort to flee from Felix. |
| E6 | (SS) | John fled from Fido, causing Fido to bite John. |
| E7 | (FA) | Mort bit Felix, causing Mort to flee from Felix. |

In these episodes Jane, John, and Fred are people, Spot, Fido and Rover are dogs, Felix is a cat, and Mort is a mouse. All of these are *objects*, represented by *token* vectors. Tokens of the same type are considered to be similar to each other, but not to tokens of other types. Bite, flee, and cause are *relations*. The argument structure of the cause relation, and the patterns in which objects fill multiple roles constitutes the *higher-order* structure.

The second column classifies the relationship between each episode and the probe using Gentner et al's types of similarity: **LS** (Literal Similarity) shares relations, object features, and higher-order structure; **AN** (Analogy, also called True Analogy) shares relations and higher-order structure, but not object features; **SS** (Surface Similarity, also called Mere Appearance) shares relations and object features, but not higher-order structure; **FA** (False Analogy) shares relations only. $AN^{cm}$ denotes a *cross-mapped* analogy – it involves the same types of objects as the probe, but the types of corresponding objects are swapped.

## 2.2  MAC/FAC PERFORMANCE ON TEST EXAMPLES

The first stage of MAC/FAC (the "Many Are Called" stage) only inspects object features and relations. It uses a vector representation of surface features. Each location in the vector corresponds to a surface feature of an object, relation or function, and the value in the location is the number of times the feature occurs in the structure. The first-stage estimate of the similarity between two structures is the dot-product of their feature-count vectors. A threshold is used to select likely analogies. It would give **E1** (LS), **E2** ($AN^{cm}$), and **E6** (SS) equal and highest scores, i.e., $(LS, AN^{cm}, SS) > (AN, FA)$

The Structure Mapping Engine (SME) (Falkenhainer, Forbus and Gentner, 1989) is used as the second stage of MAC/FAC (the "Few Are Chosen" stage). The rules of SME are that mapped relations must match, all the arguments of mapped relations must be mapped consistently, and mapping of objects must be one-to-one. SME would detect structural correspondences between each episode and the probe and give the literally similar and analogous episodes the highest rankings, i.e., $LS > AN > (SS, FA)$.

A simplified view of the overall similarity scores from MAC and the full MAC/FAC is shown in Table 1. There are four conditions – the two structures being compared can be similar in structure and/or in object attributes. In all four conditions, the structures are assumed to involve similar relations – only structural and object attribute similarities are varied. Ideally, the responses to the mixed conditions should be flexible, and controlled by which aspects of similarity are currently considered important. Only the relative values of the scores are important, the absolute values do not matter.

| Structural Similarity | Object Attribute Similarity | | Structural Similarity | Object Attribute Similarity | |
|---|---|---|---|---|---|
| | YES | NO | | YES | NO |
| YES | (LS)  High | (AN)  Low | YES | (LS)    High | (AN) ↕Med-High |
| NO | (SS)  High | (FA)  Low | NO | (SS) ↕Med-Low | (FA)    Low |

(a) Scores from MAC.                    (b) Ideal similarity scores.

Table 1: (a) Scores from the fast (MAC) similarity estimator in MAC/FAC. (b) Scores from an ideal structure-sensitive similarity estimator, e.g., SME as used in MAC/FAC.

In the remainder of this paper I describe how HRRs can be used to compute fast similarity estimates that are more like ratings in Table 1b, i.e., estimates that are flexible and sensitive to structure.

# 3   HOLOGRAPHIC REDUCED REPRESENTATIONS

A distributed representation for nested relational structures requires a solution to the binding problem. The representation of a relation such as bite(spot,jane) ("Spot bit Jane.") must bind 'Spot' to the agent role and 'Jane' to the object role. In order to represent nested structures it must also be possible to bind a relation to a role, e.g., bite(spot,jane) and the antecedent role of the cause relation.

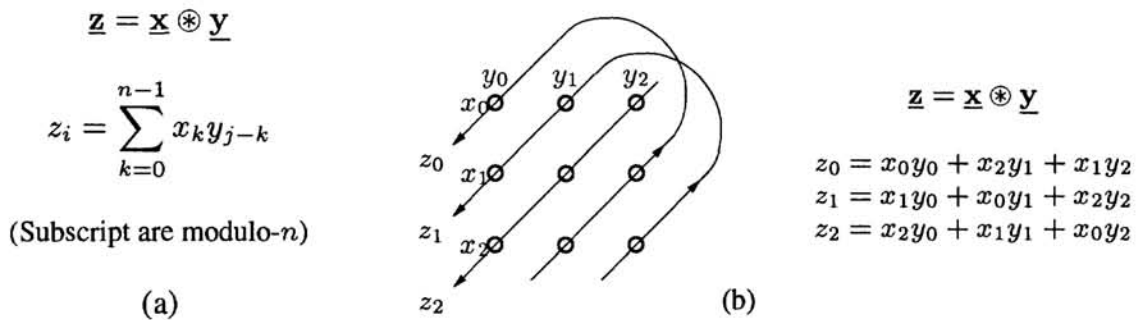

$$\underline{\mathbf{z}} = \underline{\mathbf{x}} \circledast \underline{\mathbf{y}}$$

$$z_i = \sum_{k=0}^{n-1} x_k y_{j-k}$$

(Subscript are modulo-$n$)

(a)

$$\underline{\mathbf{z}} = \underline{\mathbf{x}} \circledast \underline{\mathbf{y}}$$

$$z_0 = x_0 y_0 + x_2 y_1 + x_1 y_2$$
$$z_1 = x_1 y_0 + x_0 y_1 + x_2 y_2$$
$$z_2 = x_2 y_0 + x_1 y_1 + x_0 y_2$$

(b)

Figure 2: (a) Circular convolution. (b) Circular convolution illustrated as a compressed outer product for $n = 3$. Each of the small circles represents an element of the outer product of $x$ and $y$, e.g., the middle bottom one is $x_2 y_1$. The elements of the circular convolution of $x$ and $y$ are the sums of the outer product elements along the wrapped diagonal lines.

Holographic Reduced Representations (HRRs) (Plate, 1994) use circular convolution to solve the binding problem. Circular convolution (Figure 2a) is an operation that maps two $n$-dimensional vectors onto one $n$-dimensional vector. It can be viewed as a compressed outer product, as shown in Figure 2b. Algebraically, circular convolution behaves like multiplication – it is commutative, associative, and distributes over addition. Circular

convolution is similarity preserving: if $\underline{\mathbf{a}} \approx \underline{\mathbf{a}}'$ then $\underline{\mathbf{a}} \circledast \underline{\mathbf{b}} \approx \underline{\mathbf{a}}' \circledast \underline{\mathbf{b}}$. Associations can be decoded using a stable approximate inverse: $\underline{\mathbf{a}}^* \circledast (\underline{\mathbf{a}} \circledast \underline{\mathbf{b}}) \approx \underline{\mathbf{b}}$ (provided that the vector elements are normally distributed with mean zero and variance $1/n$). The approximate inverse is a permutation of vector elements: $a_i^* = a_{n-i}$. The dot-product of two vectors, a similarity measure, is: $\underline{\mathbf{a}} \cdot \underline{\mathbf{b}} = \sum_{i=0}^{n-1} a_i b_i$. High dimensional vectors ($n$ in the low thousands) must be used to ensure reliable encoding and decoding.

The HRR for $\texttt{bite(spot,jane)}$ is: $\mathbf{F} = <\mathbf{bite} + \mathbf{bite}_{agt} \circledast \mathbf{spot} + \mathbf{bite}_{obj} \circledast \mathbf{jane}>$, where $< \cdot >$ is a normalization operation ($<\underline{\mathbf{a}}> = \underline{\mathbf{a}}/\sqrt{\underline{\mathbf{a}} \cdot \underline{\mathbf{a}}}$). Multiple associations are superimposed in one vector and the representations for the objects (**spot** and **jane**) can also be added into the HRR in order to make it similar to other HRRs involving Spot and Jane. The HRR for a relation is the same size as the representation for an object and can be used as the filler for a role in another relation.

## 4    EXPT. 1: HRR DOT-PRODUCT SIMILARITY ESTIMATES

Experiment 1 illustrates the ways in which the dot-products of ordinary HRRs reflect, and fail to reflect, the similarity of the underlying structure of the episodes.

| Base vectors | Token vectors | |
|---|---|---|
| person, dog, cat, mouse | jane $=<$ person $+\ \mathbf{id}_{jane}>$ | spot $=<$ dog $+\ \mathbf{id}_{spot}>$ |
| bite, flee, cause | john $=<$ person $+\ \mathbf{id}_{john}>$ | fido $=<$ dog $+\ \mathbf{id}_{fido}>$ |
| bite$_{agt}$, flee$_{agt}$, cause$_{antc}$ | fred $=<$ person $+\ \mathbf{id}_{fred}>$ | rover $=<$ dog $+\ \mathbf{id}_{rover}>$ |
| bite$_{obj}$, flee$_{from}$, cause$_{cnsq}$ | mort $=<$ mouse $+\ \mathbf{id}_{mort}>$ | felix $=<$ cat $+\ \mathbf{id}_{felix}>$ |

The set of base and tokens vectors used in Experiments 1, 2 and 3 is shown above. All base and **id** vectors had elements independently chosen from a zero-mean normal distribution with variance $1/n$. The HRR for the probe is constructed as follows, and the HRRs for the other episodes are constructed in the same manner.

$\mathbf{P}_{bite} = <\mathbf{bite} + \mathbf{bite}_{agt} \circledast \mathbf{spot} + \mathbf{bite}_{obj} \circledast \mathbf{jane}>$
$\mathbf{P}_{flee} = <\mathbf{flee} + \mathbf{flee}_{agt} \circledast \mathbf{jane} + \mathbf{flee}_{from} \circledast \mathbf{spot}>$
$\mathbf{P}_{objects} = <\mathbf{jane} + \mathbf{spot}>$
$\mathbf{P} = <\mathbf{cause} + \mathbf{P}_{objects} + \mathbf{P}_{bite} + \mathbf{P}_{flee} + \mathbf{cause}_{antc} \circledast \mathbf{P}_{bite} + \mathbf{cause}_{cnsq} \circledast \mathbf{P}_{flee}>$

Experiment 1 was run 100 times, each time with a new choice of random base vectors. The vector dimension was 2048. The means and standard deviations of the HRR dot-products of the probe and each episode are shown in Table 2.

| | | | Dot-product with probe | | | |
|---|---|---|---|---|---|---|
| Probe: | | Spot bit Jane, causing Jane to flee from Spot. | Expt1 | | Expt2 | Expt3 |
| Episodes in long-term memory: | | | Avg | Sd | | |
| E1 | LS | Fido bit John, causing John to flee from Fido. | 0.70 | 0.016 | 0.63 | 0.81 |
| E2 | AN$^{cm}$ | Fred bit Rover, causing Rover to flee from Fred. | 0.47 | 0.022 | 0.47 | 0.69 |
| E3 | AN | Felix bit Mort, causing Mort to flee from Felix. | 0.39 | 0.024 | 0.39 | 0.61 |
| E6 | SS | John fled from Fido, causing Fido to bite John. | 0.47 | 0.018 | 0.44 | 0.53 |
| E7 | FA | Mort bit Felix, causing Mort to flee from Felix. | 0.39 | 0.024 | 0.39 | 0.39 |

Table 2: Results of Experiments 1, 2 and 3.

In 94 out of 100 runs, the ranking of the HRR dot-products was consistent with

$$\text{LS} > (\text{AN}^{cm}, \text{SS}) > (\text{FA}, \text{AN})$$

(where the ordering within the parenthesis varies). The order violations are due to "random" fluctuations of dot-products, whose variance decreases as the vector dimension increases. When the experiment was rerun with vector dimension 4096 there was only one violation of this order out of 100 runs.

These results represent an improvement over the first stage of MAC/FAC – the HRR dot-product distinguishes between literal and surface similarity. However, when the episodes do not share object attributes, the HRR dot-product is not affected by structural similarity and the scores do not distinguish analogy from false analogy or superficial similarity.

# 5   EXPERIMENTS 2 AND 3: CONTEXTUALIZED HRRS

Dot-product comparisons of HRRs are not sensitive to structural similarity in the absence of similar objects. This is because the way in which objects fill multiple roles is not expressed as a surface feature in HRRs. Consequently, the analogous episodes $\mathbf{E2}$ ($AN^{cm}$) and $\mathbf{E3}$ ($AN$) do not receive higher scores than the non analogous episodes $\mathbf{E6}$ (SS) and $\mathbf{E7}$ (FA).

We can force role structure to become a surface feature by "contextualizing" the representations of fillers. Contextualization involves incorporating information about what other roles an object fills in the representation of a filler. This is like thinking of Spot (in the probe) as an entity that bites (a biter) and an entity that is fled from (a "fled-from").

In ordinary HRRs the filler alone is convolved with the role. In contextualized HRRs a blend of the filler and its context is convolved with the role. The representation for the context of object in a role is the typical fillers of the *other* roles the object fills. The context for Spot in the flee relation is represented by $\mathbf{typ}_{agt}^{bite}$ and the context in the bite relation is represented by $\mathbf{typ}_{from}^{flee}$ (where $\mathbf{typ}_{agt}^{bite} = \mathbf{bite} \circledast \mathbf{bite}_{agt}^*$ and $\mathbf{typ}_{from}^{flee} = \mathbf{flee} \circledast \mathbf{flee}_{from}^*$). The degree of contextualization is governed by the mixing proportions $\kappa_o$ (object) and $\kappa_c$ (context). The contextualized HRR for the probe is constructed as follows:

$$\mathbf{P}_{bite} = < \mathbf{bite} + \mathbf{bite}_{agt} \circledast (\kappa_o \mathbf{spot} + \kappa_c \mathbf{typ}_{from}^{flee}) + \mathbf{bite}_{obj} \circledast (\kappa_o \mathbf{jane} + \kappa_c \mathbf{typ}_{agt}^{flee}) >$$
$$\mathbf{P}_{flee} = < \mathbf{flee} + \mathbf{flee}_{agt} \circledast (\kappa_o \mathbf{jane} + \kappa_c \mathbf{typ}_{obj}^{bite}) + \mathbf{flee}_{from} \circledast (\kappa_o \mathbf{spot} + \kappa_c \mathbf{typ}_{agt}^{bite}) >$$
$$\mathbf{P}_{objects} = < \mathbf{jane} + \mathbf{spot} >$$
$$\mathbf{P} = < \mathbf{cause} + \mathbf{P}_{objects} + \mathbf{P}_{bite} + \mathbf{P}_{flee} + \mathbf{cause}_{antc} \circledast \mathbf{P}_{bite} + \mathbf{cause}_{cnsq} \circledast \mathbf{P}_{flee} >$$

A useful similarity estimator must be flexible and able to adjust salience of different aspects of similarity according to context or command. The degree to which role-alignment affects the HRR dot-product can be adjusted by changing the degree of contextualization in just one episode of a pair. Hence, the items in memory can be encoded with a fixed $\kappa$ values ($\kappa_o^m$ and $\kappa_c^m$) and the salience of role alignment can be changed by altering the degree of contextualization in the probe ($\kappa_o^p$ and $\kappa_c^p$). This is fortunate as it would be impractical to recode all items in memory in order to alter the salience of role alignment in a particular comparison. The same technique can be used to adjust the importance of other features.

Two experiments were performed with contextualized HRRs, with the same episodes as used in Experiment 1. In Experiment 2 the probe was non-contextualized ($\kappa_o^p = 1, \kappa_c^p = 0$), and in Experiment 3 the probe was contextualized ($\kappa_o^p = 1/\sqrt{2}, \kappa_c^p = 1/\sqrt{2}$). For both Experiments 2 and 3 the episodes in memory were encoded with the same degree of contextualization ($\kappa_o^m = 1/\sqrt{2}, \kappa_c^m = 1/\sqrt{2}$). As before, each set of comparisons was run 100 times, and the vector dimension was 2048. The results are shown in Table 2.

The scores in Experiment 2 (non-contextualized probe) were consistent (in 95 out of 100 runs) with the same order as given for Experiment 1:

$$LS > (AN^{cm}, SS) > (FA, AN)$$

The scores in Experiment 3 (contextualized probe) were consistent (in all 100 runs) with an ordering that ranks analogous episodes as strictly more similar than non-analogous ones:

$$LS > AN^{cm} > AN > SS > FA$$

# 6    DISCUSSION

The dot-product of HRRs provides a fast estimate of the degree of analogical match and is sensitive to various structural aspects of the match. It is not intended to be a model of complex or creative analogy making, but it could be a useful first stage in a model of analogical reminding.

| Structural Similarity | Object Attribute Similarity | |
|---|---|---|
| | YES | NO |
| YES | (LS)  High | (AN)  Low |
| NO | (SS)  Med | (FA)  Low |

(a) Ordinary-HRR dot-products.

| Structural Similarity | Object Attribute Similarity | |
|---|---|---|
| | YES | NO |
| YES | (LS)        High | (AN) $\updownarrow$Med-High |
| NO | (SS) $\updownarrow$Med-Low | (FA)        Low |

(b) Contextualized-HRR dot-products.

Table 3: Similarity scores from ordinary and contextualized HRR dot-product comparisons. The flexibility comes adjusting the weights of various components in the probe.

The dot-product of ordinary HRRs is sensitive to some aspects of structural similarity. It improves on the existing fast similarity matcher in MAC/FAC in that it discriminates the first column of Table 3 – it ranks literally similar (LS) episodes higher than superficially similar (SS) episodes. However, it is insensitive to structural similarity when corresponding objects are not similar. Consequently, it ranks both analogies (AN) and false analogies (FA) lower than superficially similar (SS) episodes.

The dot-product of contextualized HRRs is sensitive to structural similarity even when corresponding objects are not similar. It ranks the given examples in the same order as would the full MAC/FAC or ARCS system.

Contextualization does not cause all relational structure to be expressed as surface features in the HRR vector. It only suffices to distinguish analogous from non-analogous structures when no two entities fill the same set of roles. Sometimes, the distinguishing context for an object is more than the other roles that the object fills. Consider the situation where two boys are bitten by two dogs, and each flees from the dog that did not bite him. With contextualization as described above it is impossible to distinguish this from the situation where each boy flees from the dog that did bite him.

HRR dot-products are flexible – the salience of various aspects of similarity can be adjusted by changing the weights of various components in the probe. This is true for both ordinary and contextualized HRRs.

HRRs retain many of the advantages of ordinary distributed representations: (a) There is a simple and computationally efficient measure of similarity between two representations –

the vector dot-product. Similar items can be represented by similar vectors. (b) Items are represented in a continuous space. (c) Information is distributed and redundant.

Hummel and Biederman (1992) discussed the binding problem and identified two main problems faced by conjunctive coding approaches such as Tensor Products (Smolensky, 1990). These are exponential growth of the size of the representation with the number of associated objects (or attributes), and insensitivity to attribute structure. HRRs have much in common with conjunctive coding approaches (they can be viewed as a compressed conjunctive code), but do not suffer from these problems. The size of HRRs remains constant with increasing numbers of associated objects, and sensitivity to attribute structure has been demonstrated in this paper.

The HRR dot-product is not without its drawbacks. Firstly, examples for which it will produce counter-intuitive rankings can be constructed. Secondly, the scaling with the size of episodes could be a problem – the sum of structural-feature matches becomes a less appropriate measure of similarity as the episodes get larger. A possible solution to this problem is to construct a spreading activation network of HRRs in which each episode is represented as a number of chunks, and each chunk is represented by a node in the network.

The software used for the HRR calculations is available from the author.

## Footnotes

[1]"Surface features" of stories are the features of the entities and relations involved, and "structural features" are the relationships among the relations and entities.

## References

Falkenhainer, B., Forbus, K. D., and Gentner, D. (1989). The Structure-Mapping Engine: Algorithm and examples. *Artificial Intelligence*, 41:1–63.

Gentner, D. and Forbus, K. D. (1991). MAC/FAC: A model of similarity-based retrieval. In *Proceedings of the Thirteenth Annual Cognitive Science Society Conference*, pages 504–509, Hillsdale, NJ. Erlbaum.

Gentner, D. and Markman, A. B. (1992). Analogy – Watershed or Waterloo? Structural alignment and the development of connectionist models of analogy. In Giles, C. L., Hanson, S. J., and Cowan, J. D., editors, *Advances in Neural Information Processing Systems 5 (NIPS*92)*, pages 855–862, San Mateo, CA. Morgan Kaufmann.

Gentner, D., Rattermann, M. J., and Forbus, K. D. (1993). The roles of similarity in transfer: Separating retrievability from inferential soundness. *Cognitive Psychology*, 25:431–467.

Hummel, J. E. and Biederman, I. (1992). Dynamic binding in a neural network for shape recognition. *Psychological Review*, 99(3):480–517.

Markman, A. B., Gentner, D., and Wisniewski, E. J. (1993). Comparison and cognition: Implications of structure-sensitive processing for connectionist models. Unpublished manuscript.

Plate, T. A. (1991). Holographic Reduced Representations: Convolution algebra for compositional distributed representations. In Mylopoulos, J. and Reiter, R., editors, *Proceedings of the 12th International Joint Conference on Artificial Intelligence*, pages 30–35, San Mateo, CA. Morgan Kaufmann.

Plate, T. A. (1993). Estimating analogical similarity by vector dot-products of Holographic Reduced Representations. Unpublished manuscript.

Plate, T. A. (1994). Holographic reduced representations. *IEEE Transactions on Neural Networks*. To appear.

Smolensky, P. (1990). Tensor product variable binding and the representation of symbolic structures in connectionist systems. *Artificial Intelligence*, 46(1-2):159–216.

Thagard, P., Holyoak, K. J., Nelson, G., and Gochfeld, D. (1990). Analog Retrieval by Constraint Satisfaction. *Artificial Intelligence*, 46:259–310.

Wharton, C. M., Holyoak, K. J., Downing, P. E., Lange, T. E., Wickens, T. D., and Melz, E. R. (1994). Below the surface: Analogical similarity and retrieval competition in reminding. *Cognitive Psychology*. To appear.
